# Quantification and the language of thought

**Charles Kemp**
Department of Psychology
Carnegie Mellon University
ckemp@cmu.edu

## Abstract

Many researchers have suggested that the psychological complexity of a concept is related to the length of its representation in a language of thought. As yet, however, there are few concrete proposals about the nature of this language. This paper makes one such proposal: the language of thought allows first order quantification (quantification over objects) more readily than second-order quantification (quantification over features). To support this proposal we present behavioral results from a concept learning study inspired by the work of Shepard, Hovland and Jenkins.

Humans can learn and think about many kinds of concepts, including natural kinds such as elephant and water and nominal kinds such as grandmother and prime number. Understanding the mental representations that support these abilities is a central challenge for cognitive science. This paper proposes that quantification plays a role in conceptual representation—for example, an animal $X$ qualifies as a predator if there is *some* animal $Y$ such that $X$ hunts $Y$. The concepts we consider are much simpler than real-world examples such as predator, but even simple laboratory studies can provide important clues about the nature of mental representation.

Our approach to mental representation is based on the language of thought hypothesis [1]. As pursued here, the hypothesis proposes that mental representations are constructed in a compositional language of some kind, and that the psychological complexity of a concept is closely related to the length of its representation in this language [2, 3, 4]. Following previous researchers [2, 4], we operationalize the psychological complexity of a concept in terms of the ease with which it is learned and remembered. Given these working assumptions, the remaining challenge is to specify the representational resources provided by the language of thought. Some previous studies have relied on propositional logic as a representation language [2, 5], but we believe that the resources of predicate logic are needed to capture the structure of many human concepts. In particular, we suggest that the language of thought can accommodate relations, functions, and quantification, and focus here on the role of quantification.

Our primary proposal is that quantification is supported by the language of thought, but that quantification over objects is psychologically more natural than quantification over features. To test this idea we compare concept learning in two domains which are very similar except for one critical difference: one domain allows quantification over objects, and the other allows quantification over features. We consider several logical languages that can be used to formulate concepts in both domains, and find that learning times are best predicted by a language that supports quantification over objects but not features.

Our work illustrates how theories of mental representation can be informed by comparing concept learning across two or more domains. Existing studies work with a range of domains, and it is useful to consider a "conceptual universe" that includes these possibilities along with many others that have not yet been studied. Table 1 charts a small fragment of this universe, and the penultimate column shows example stimuli that will be familiar from previous studies of concept learning. Previous studies have made important contributions by choosing a single domain in Table 1 and explaining

why some concepts within this domain are easier to learn than others [2, 4, 6, 7, 8, 9]. Comparisons across domains can also provide important information about learning and mental representation, and we illustrate this claim by comparing learning times across Domains 3 and 4.

The next section introduces the conceptual universe in Table 1 in more detail. We then present a formal approach to concept learning that relies on a logical language and compare three candidate languages. Language $OQ$ (for object quantification) supports quantification over objects but not features, language $FQ$ (for feature quantification) supports quantification over features but not objects, and language $OQ + FQ$ supports quantification over both objects and features. We use these languages to predict learning times across Domains 3 and 4, and present an experiment which suggests that language $OQ$ comes closest to the language of thought.

# 1   The conceptual universe

Table 1 provides an organizing framework for thinking about the many domains in which learning can occur. The table includes 8 domains, each of which is defined by specifying some number of objects, features, and relations, and by specifying the range of each feature and each relation. We refer to the elements in each domain as *items*, and the penultimate column of Table 1 shows items from each domain. The first row shows a domain commonly used by studies of Boolean concept learning. Each item in this domain includes a single object $a$ and specifies whether that object has value $v_1$ (small) or $v_2$ (large) on feature $F$ (size), value $v_3$ (white) or $v_4$ (gray) on feature $G$ (color), and value $v_5$ (vertical) or $v_6$ (horizontal) on feature $H$ (texture). Domain 2 also includes three features, but now each item includes three objects and each feature applies to only one of the objects. For example, feature $H$ (texture) applies to only the third object in the domain (i.e. the third square on each card). Domain 3 is similar to Domain 1, but now the three features can be aligned— for any given item each feature will be absent (value 0) or present. The example in Table 1 uses three features (boundary, dots, and slash) that can each be added to an unadorned gray square. Domain 4 is similar to Domain 2, but again the feature values can be aligned, and the feature for each object will be absent (value 0) or present. Domains 5 and 6 are similar to domains 2 and 4 respectively, but each one includes relations rather than features. In Domain 6, for example, the relation $R$ assigns value 0 (absent) or value 1 (present) to each undirected pair of objects.

The first six domains in Table 1 are all variants of Domain 1, which is the domain typically used by studies of Boolean concept learning. Focusing on six related domains helps to establish some of the dimensions along which domains can differ, but the final two domains in Table 1 show some of the many alternative possibilities. Domain 7 includes two categorical features, each of which takes three rather than two values. Domain 8 is similar to Domain 6, but now the number of objects is 6 rather than 3 and relation $R$ is directed rather than undirected. To mention just a handful of possibilities which do not appear in Table 1, domains may also have categorical features that are ordered (e.g. a size feature that takes values small, medium, and large), continuous valued features or relations, relations with more than two places, and objects that contain sub-objects or parts.

Several learning problems can be formulated within any given domain. The most basic is to learn a single item—for example, a single item from Domain 8 [4]. A second problem is to learn a class of items—for example, a class that includes four of the items in Domain 1 and excludes the remaining four [6]. Learning an item class can be formalized as learning a unary predicate defined over items, and a natural extension is to consider predicates with two or more arguments. For example, problems of the form $A$ is to $B$ as $C$ is to ? can be formulated as problems where the task is to learn a binary relation analogous$(\cdot, \cdot)$ given the single example analogous$(A, B)$. Here, however, we focus on the task of learning item classes or unary predicates.

Since we focus on the role of quantification, we will work with domains where quantification is appropriate. Quantification over objects is natural in cases like Domain 4 where the feature values for all objects can be aligned. Note, for example, that the statement "every object has its feature" picks out the final example item in Domain 4 but that no such statement is possible in Domain 2. Quantification over features is natural in cases like Domain 3 where the ranges of each feature can be aligned. For example, "object $a$ has all three features" picks out the final example item in Domain 3 but no such statement is possible in Domain 1. We therefore focus on Domains 3 and 4, and explore the problem of learning item classes in each domain.

| # | Objects $O$ | Domain specification Features | Relations | Example Items | Ref. |
|---|---|---|---|---|---|
| 1 | $\{a\}$ | $F : O \rightarrow \{v_1, v_2\}$<br>$G : O \rightarrow \{v_3, v_4\}$<br>$H : O \rightarrow \{v_5, v_6\}$ | — |  | [2, 6, 7, 10, 11] |
| 2 | $\{a, b, c\}$ | $F : a \rightarrow \{v_1, v_2\}$<br>$G : b \rightarrow \{v_3, v_4\}$<br>$H : c \rightarrow \{v_5, v_6\}$ | — |  | [6] |
| 3 | $\{a\}$ | $F : O \rightarrow \{0, v_1\}$<br>$G : O \rightarrow \{0, v_2\}$<br>$H : O \rightarrow \{0, v_3\}$ | — |  | [12] |
| 4 | $\{a, b, c\}$ | $F : a \rightarrow \{0, v_1\}$<br>$G : b \rightarrow \{0, v_2\}$<br>$H : c \rightarrow \{0, v_3\}$ | — |  | [6] |
| 5 | $\{a, b, c\}$ | — | $R : (a, b) \rightarrow \{v_1, v_2\}$<br>$S : (a, c) \rightarrow \{v_3, v_4\}$<br>$T : (b, c) \rightarrow \{v_5, v_6\}$ |  | |
| 6 | $\{a, b, c\}$ | — | $R : O \times O \rightarrow \{0, 1\}$ |  | [13] |
| 7 | $\{a\}$ | $F : O \rightarrow \{v_1, v_2, v_3\}$<br>$G : O \rightarrow \{v_4, v_5, v_6\}$ | — |  | [8, 9] |
| 8 | $\{a, b, c, d, e, f\}$ | — | $R : O \times O \rightarrow \{0, 1\}$ |  | [4] |

Table 1: The conceptual universe. Eight domains are shown, and each one is defined by a set of objects, a set of features, and a set of relations. We call the members of each domain *items*, and an item is created by specifying the extension of each feature and relation in the domain. The six domains above the double lines are closely related to the work of Shepard et al. [6]. Each one includes eight items which differ along three dimensions. These dimensions, however, emerge from different underlying representations in the six cases.

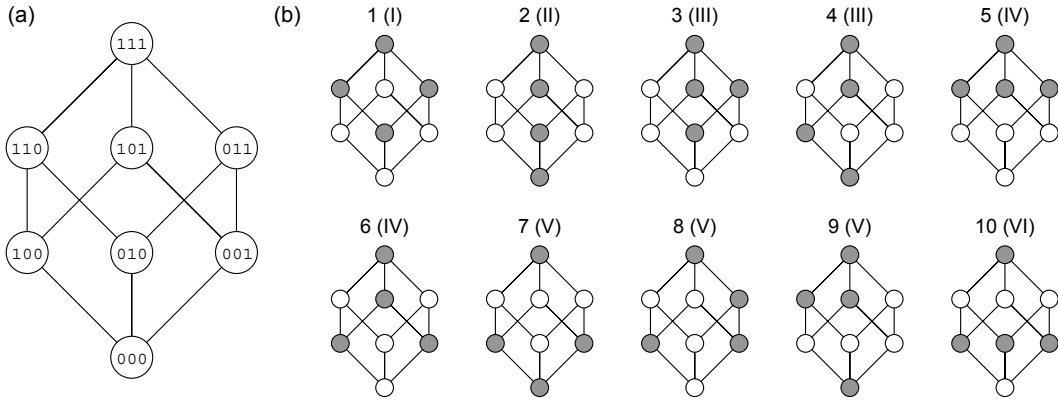

Figure 1: (a) A stimulus lattice for domains (e.g. Domains 3, 4, and 6) that can be encoded as a triple of binary values where `0` represents "absent" and `1` represents "present." (b) If the order of the values in the triple is not significant, there are 10 distinct ways to partition the lattice into two classes of four items. The SHJ type for each partition is shown in parentheses.

Domains 3 and 4 both include 8 items each and we will consider classes that include exactly four of these items. Each item in these domains can be represented as a triple of binary values, where 0 indicates that a feature is absent and value 1 indicates that a feature is present. Each triple represents the values of the three features (Domain 3) or the feature values for the three objects (Domain 4). By representing each domain in this way, we have effectively adopted domain specifications that are simplifications of those shown in Table 1. Domain 3 is represented using three features of the form $F, G, H : O \rightarrow \{0, 1\}$, and Domain 4 is represented using a single feature of the form $F : O \rightarrow \{0, 1\}$. Simplifications of this kind are possible because the features in each domain can be aligned—notice that no corresponding simplifications are possible for Domains 1 and 2.

The eight binary triples in each domain can be organized into the lattice shown in Figure 1a. Here we consider all ways to partition the vertices of the lattice into two groups of four. If partitions that differ only up to a permutation of the features (Domain 3) or objects (Domain 4) are grouped into equivalence classes, there are ten of these classes, and a representative of each is shown in Figure 1b. Previous researchers [6] have pointed out that the stimuli in Domain 1 can be organized into a cube similar to Figure 1a, and that there are six ways to partition these stimuli into two groups of four up to permutations of the features and permutations of the range of each feature. We refer to these equivalence classes as the six Shepard-Hovland-Jenkins types (or SHJ types), and each partition in Figure 1b is labeled with its corresponding SHJ type label. Note, for example, that partitions 3 and 4 are both examples of SHJ type III. For us, partitions 3 and 4 are distinct since items `000` (all absent) and `111` (all present) are uniquely identifiable, and partition 3 assigns these items to different classes but partition 4 does not.

Previous researchers have considered differences between some of the first six domains in Table 1. Shepard et al. [6] ran experiments using *compact* stimuli (Domain 1) and *distributed* stimuli (Domains 2 and 4), and observed the same difficulty ranking of the six SHJ types in all cases. Their work, however, does not acknowledge that Domain 4 leads to 10 distinct types rather than 6, and therefore fails to address issues such as the relative complexities of concepts 5 and 6 in Figure 1. Social psychologists [13, 14] have studied Domain 6 and found that learning patterns depart from the standard SHJ order—in particular, that SHJ type VI (Concept 10 in Figure 1) is simpler than types III, IV and V. This finding has been used to support the claim that social learning relies on a domain-specific principle of *structural balance* [14]. We will see, however, that the relative simplicity of type VI in domains like 4 and 6 is consistent with a domain-general account based on representational economy.

## 2 A representation length approach to concept learning

The conceptual universe in Table 1 calls for an account of learning that can apply across many domains. One candidate is the representation length approach, which proposes that concepts are mentally represented in a language of thought, and that the subjective complexity of a concept is

determined by the length of its representation in this language [4]. We consider the case where a concept corresponds to a class of items, and explore the idea that these concepts are mentally represented in a logical language. More formally, a concept is represented as a logical sentence, and the concept includes all *models* of this sentence, or all items that make the sentence true.

The predictions of this representation length approach depend critically on the language chosen. Here we consider three languages—an *object quantification* language $OQ$ that supports quantification over objects, a *feature quantification* language $FQ$ that supports quantification over features, and a language $OQ + FQ$ that supports quantification over both objects and features. Language $OQ$ is based on a standard logical language known as predicate logic with equality. The language includes symbols representing objects (e.g. a and b), and features (e.g. F and G) and these symbols can be combined to create *literals* that indicate that an object does ($F_a$) or does not have a certain feature ($F_a'$). Literals can be combined using two connectives: AND ($F_aG_a$) and OR ($F_a + G_a$). The language includes two quantifiers—for all ($\forall$) and there exists ($\exists$)—and allows quantification over objects (e.g. $\forall_x F_x$, where x is a variable that ranges over all objects in the domain). Finally, language $OQ$ includes equality and inequality relations ($=$ and $\neq$) which can be used to compare objects and object variables (e.g. $=_{xa}$ or $\neq_{xy}$).

Table 2 shows several sentences formulated in language $OQ$. Suppose that the $OQ$ complexity of each sentence is defined as the number of basic propositions it contains, where a basic proposition can be a positive or negative literal ($F_a$ or $F_a'$) or an equality or inequality statement ($=_{xa}$ or $\neq_{xy}$). Equivalently, the complexity of a sentence is the total number of ANDs plus the total number of ORs plus one. This measure is equivalent by design to Feldman's [2] notion of Boolean complexity when applied to a sentence without quantification. The complexity values in Table 2 show minimal complexity values for each concept in Domains 3 and 4. Table 2 also shows a single sentence that achieves each of these complexity values, although some concepts admit multiple sentences of minimal complexity.

The complexity values in Table 2 were computed using an "enumerate then combine" approach. We began by enumerating a set of sentences according to criteria described in the next paragraph. Each sentence has an extension that specifies which items in the domain are consistent with the sentence. Given the extensions of all sentences generated during the enumeration phase, the combination phase considered all possible ways to combine these extensions using conjunctions or disjunctions. The procedure terminated once extensions corresponding to all of the concepts in the domain had been found. Although the number of possible sentences grows rapidly as the complexity of these sentences increases, the number of extensions is fixed and relatively small ($2^8$ for domains of size 8). The combination phase is tractable since sentences with the same extension can be grouped into a single equivalence class.

The enumeration phase considered all formulae which had at most two quantifiers and which had a complexity value lower than four. For example, this phase did not include the formula $\exists_x\exists_y\exists_z \neq_{yz} F_x'F_yF_z$ (too many quantifiers) or the formula $\forall_x\exists_y \neq_{xy} F_y(F_x + G_x + H_x)$ (complexity too high). Despite these restrictions, we believe that the complexity values in Table 2 are identical to the values that would be obtained if we had considered all possible sentences.

Language $FQ$ is similar to $OQ$ but allows quantification over features rather than objects. For example, $FQ$ includes the statement $\forall_Q Q_a$, where Q is a variable that ranges over all features in the domain. Language $FQ$ also allows features and feature variables to be compared for equality or inequality (e.g. $=_{QF}$ or $\neq_{QR}$). Since $FQ$ and $OQ$ are closely related, it follows that the $FQ$ complexity values for Domains 3 and 4 are identical to the $OQ$ complexity values for Domains 4 and 3. For example, $FQ$ can express concept 5 in Domain 3 as $\forall_Q\exists_R \neq_{QR} R_a$.

We can combine $OQ$ and $FQ$ to create a language $OQ + FQ$ that allows quantification over both objects and features. Allowing both kinds of quantification leads to identical complexity values for Domains 3 and 4. Language $OQ + FQ$ can express each of the formulae for Domain 4 in Table 2, and these formulae can be converted into corresponding formulae for Domain 3 by translating each instance of object quantification into an instance of feature quantification.

Logicians distinguish between first-order logic, which allows quantification over objects but not predicates, and second-order logic, which allows quantification over objects and predicates. The difference between languages $OQ$ and $OQ + FQ$ is superficially similar to the difference between first-order and second-order logic, but does not cut to the heart of this matter. Since language

| # | Domain 3 | $C$ | Domain 4 | $C$ |
|---|---|---|---|---|
| 1 | $G_a$ | 1 | $F_b$ | 1 |
| 2 | $F_a'H_a' + F_aH_a$ | 4 | $F_a'F_c' + F_aF_c$ | 4 |
| 3 | $F_a'G_a + F_aH_a$ | 4 | $F_a'F_b + F_aF_c$ | 4 |
| 4 | $F_a'G_a' + F_aH_a$ | 4 | $F_a'F_b' + F_aF_c$ | 4 |
| 5 | $G_a(F_a + H_a) + F_aH_a$ | 5 | $\forall_x\exists_y \neq_{xy} F_y$ | 2 |
| 6 | $G_a'(F_a + H_a) + F_aH_a$ | 5 | $(\forall_xF_x) + F_b'\exists_yF_y$ | 3 |
| 7 | $G_a'(F_a + H_a) + F_aG_aH_a$ | 6 | $(\forall_xF_x) + F_b'(F_a' + F_c')$ | 4 |
| 8 | $H_a(F_a' + G_a) + F_aG_a'H_a'$ | 6 | $F_c(F_a' + F_b) + F_aF_b'F_c'$ | 6 |
| 9 | $F_a(G_a + H_a) + F_a'G_a'H_a'$ | 6 | $(\forall_xF_x') + F_a(F_b + F_c)$ | 4 |
| 10 | $G_a'(F_aH_a' + F_a'H_a) + G_a(F_a'H_a' + F_aH_a)$ | 10 | $(\forall_xF_x) + \exists_y\forall_zF_y(=_{zy} +F_z')$ | 4 |

Table 2: Complexity values $C$ and corresponding formulae for language $OQ$. Boolean complexity predicts complexity values for both domains that are identical to the $OQ$ complexity values shown here for Domain 3. Language $FQ$ predicts complexity values for Domains 3 and 4 that are identical to the $OQ$ values for Domains 4 and 3 respectively. Language $OQ + FQ$ predicts complexity values for both domains that are identical to the $OQ$ complexity values for Domain 4.

$OQ + FQ$ only supports quantification over a pre-specified set of features, it is equivalent to a typed first order logic that includes types for objects and features [15]. Future studies, however, can explore the cognitive relevance of higher-order logic as developed by logicians.

## 3 Experiment

Now that we have introduced languages $OQ$, $FQ$ and $OQ + FQ$ our theoretical proposals can be sharply formulated. We suggest that quantification over objects plays an important role in mental representations, and predict that $OQ$ complexity will account better for human learning than Boolean complexity. We also propose that quantification over objects is more natural than quantification over features, and predict that $OQ$ complexity will account better for human learning than both $FQ$ complexity and $OQ + FQ$ complexity. We tested these predictions by designing an experiment where participants learned concepts from Domains 3 and 4.

**Method.** 20 adults participated for course credit. Each participant was assigned to Domain 3 or Domain 4 and learned all ten concepts from that domain. The items used for each domain were the cards shown in Table 1. Note, for example, that each Domain 3 card showed one square, and that each Domain 4 card showed three squares. These items are based on stimuli developed by Sakamoto and Love [12].

The experiment was carried out using a custom built graphical interface. For each learning problem in each domain, all eight items were simultaneously presented on the screen, and participants were able to drag them around and organize them however they liked. Each problem had three phases. During the learning phase, the four items belonging to the current concept had red boundaries, and the remaining four items had blue boundaries. During the memory phase, these colored boundaries were removed, and participants were asked to sort the items into the red group and the blue group. If they made an error they returned to the learning phase, and could retake the test whenever they were ready. During the description phase, participants were asked to provide a written description of the two groups of cards. The color assignments (red or blue) were randomized across participants— in other words, the "red groups" learned by some participants were identical to the "blue groups" learned by others. The order in which participants learned the 10 concepts was also randomized.

**Model predictions.** The $OQ$ complexity values for the ten concepts in each domain are shown in Table 2 and plotted in Figure 2a. The complexity values in Figure 2a have been normalized so that they sum to one within each domain, and the differences of these normalized scores are shown in the final row of Figure 2a. The two largest bars in the difference plot indicate that Concepts 10 and 5 are predicted to be easier to learn in Domain 4 than in Domain 3. Language $OQ$ can express

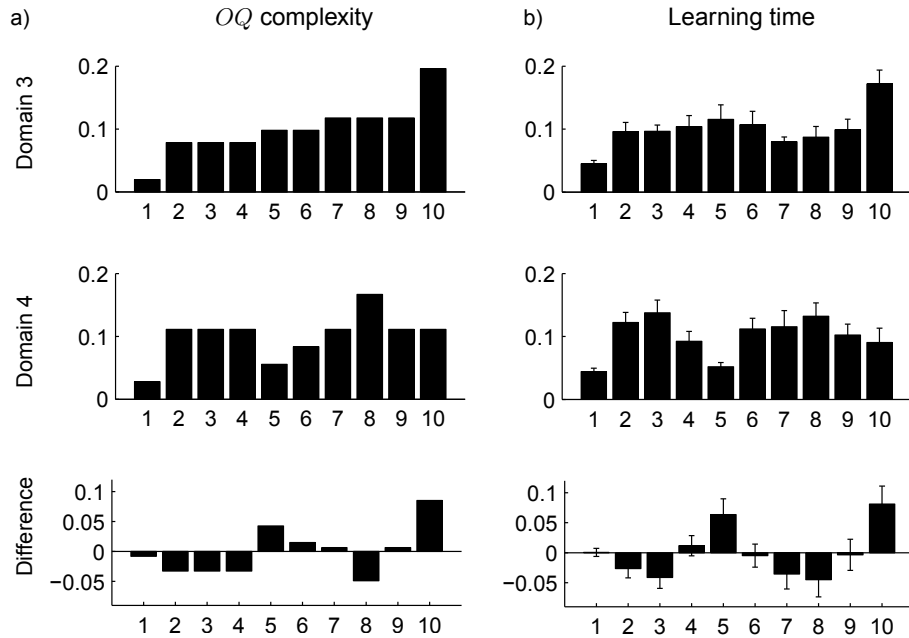

Figure 2: Normalized $OQ$ complexity values and normalized learning times for the 10 concepts in Domains 3 and 4.

statements like "either 1 or 3 objects have $F$" (Concept 10 in Domain 4), or "2 or more objects have $F$" (Concept 5 in Domain 4). Since quantification over features is not permitted, however, analogous statements (e.g. "object $a$ has either 1 or 3 features") cannot be formulated in Domain 3.

Concept 10 corresponds to SHJ type VI, which often emerges as the most difficult concept in studies of Boolean concept learning. Our model therefore predicts that the standard ordering of the SHJ types will not apply in Domain 4. Our model also predicts that concepts assigned to the same SHJ type will have different complexities. In Domain 4 the model predicts that Concept 6 will be harder to learn than Concept 5 (both are examples of SHJ type IV), and that Concept 8 will be harder to learn than Concepts 7 or 9 (all three are examples of SHJ type V).

**Results.** The computer interface recorded the amount of time participants spent on the learning phase for each concept. Domain 3 was a little more difficult than Domain 4 overall: on average, Domain 3 participants took 557 seconds and Domain 4 participants took 467 seconds to learn the 10 concepts. For all remaining analyses, we consider learning times that are normalized to sum to 1 for each participant. Figure 2b shows the mean values for these normalized times, and indicates the relative difficulties of the concepts within each condition.

The difference plot in Figure 2b supports the two main predictions identified previously. Concepts 10 and 5 are the cases that differ most across the domains, and both concepts are easier to learn in Domain 3 than Domain 4. As predicted, Concept 5 is substantially easier than Concept 6 in Domain 4 even though both correspond to the same SHJ type. Concepts 7 through 9 also correspond to the same SHJ type, and the data for Domain 4 suggest that Concept 8 is the most difficult of the three, although the difference between Concepts 8 and 7 is not especially large.

Four sets of complexity predictions are plotted against the human data in Figure 3. Boolean complexity and $OQ$ complexity make identical predictions about Domain 3, and $OQ$ complexity and $OQ + FQ$ complexity make identical predictions about Domain 4. Only $OQ$ complexity, however, accounts for the results observed in both domains.

The concept descriptions generated by participants provide additional evidence that there are psychologically important differences between Domains 3 and 4. If the descriptions for concepts 5 and 10 are combined, 18 out of 20 responses in Domain 4 referred to quantification or counting. One representative description of Concept 5 stated that "red has multiple filled" and that "blue has one filled or none." Only 3 of 20 responses in Domain 3 mentioned quantification. One representative description of Concept 5 stated that "red = multiple features" and that "blue = only one feature."

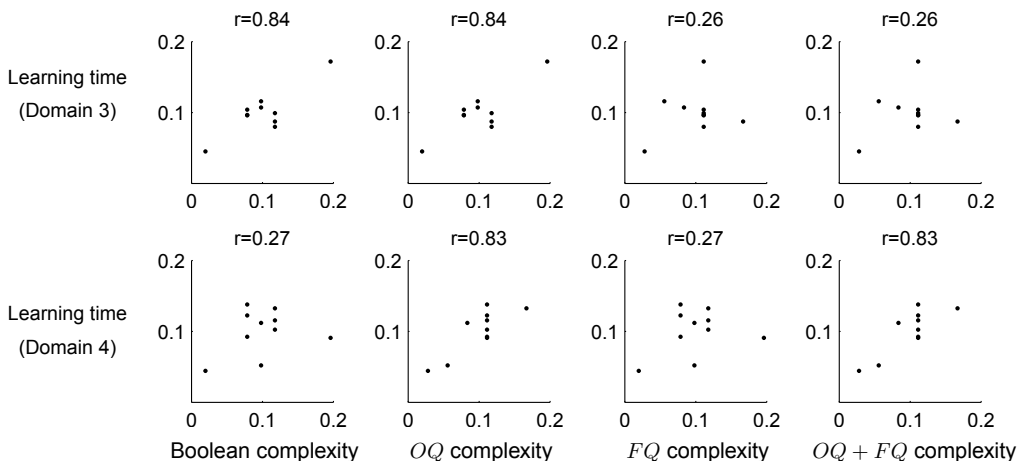

Figure 3: Normalized learning times for each domain plotted against normalized complexity values predicted by four languages: Boolean logic, $OQ$, $FQ$ and $OQ + FQ$.

These results suggest that people can count or quantify over features, but that it is psychologically more natural to quantify over objects rather than features.

Although we have focused on three specific languages, the results in Figure 2b can be used to evaluate alternative proposals about the language of thought. One such alternative is an extension of Language $OQ$ that allows feature values to be compared for equality. This extended language supports concise representations of Concept 2 in both Domain 3 ($F_a = H_a$) and Domain 4 ($F_a = F_c$), and predicts that Concept 2 will be easier to learn than all other concepts except Concept 1. Note, however, that this prediction is not compatible with the data in Figure 2b. Other languages might also be considered, but we know of no simple language that will account for our data better than $OQ$.

## 4   Conclusion

Comparing concept learning across qualitatively different domains can provide valuable information about the nature of mental representation. We compared two domains that that are similar in many respects, but that differ according to whether they include a single object (Domain 3) or multiple objects (Domain 4). Quantification over objects is possible in Domain 4 but not Domain 3, and this difference helps to explain the different learning patterns we observed across the two domains. Our results suggest that concept representations can incorporate quantification, and that quantifying over objects is more natural than quantifying over features.

The model predictions we reported are based on a language ($OQ$) that is a generic version of first order logic with equality. Our results therefore suggest that some of the languages commonly considered by logicians (e.g. first order logic with equality) may indeed capture some aspects of the "laws of thought" [16]. A simple language like $OQ$ offers a convenient way to explore the role of quantification, but this language will need to be refined and extended in order to provide a more accurate account of mental representation. For example, a comprehensive account of the language of thought will need to support quantification over features in some cases, but might be formulated so that quantification over features is typically more costly than quantification over objects.

Many possible representation languages can be imagined and a large amount of empirical data will be needed to identify the language that comes closest to the language of thought. Many relevant studies have already been conducted [2, 6, 8, 9, 13, 17], but there are vast regions of the conceptual universe (Table 1) that remain to be explored. Navigating this universe is likely to involve several challenges, but web-based experiments [18, 19] may allow it to be explored at a depth and scale that are currently unprecedented. Characterizing the language of thought is undoubtedly a long term project, but modern methods of data collection may support rapid progress towards this goal.

**Acknowledgments** I thank Maureen Satyshur for running the experiment. This work was supported in part by NSF grant CDI-0835797.

# References

[1] J. A. Fodor. *The language of thought*. Harvard University Press, Cambridge, 1975.

[2] J. Feldman. Minimization of Boolean complexity in human concept learning. *Nature*, 407: 630–633, 2000.

[3] D. Fass and J. Feldman. Categorization under complexity: A unified MDL account of human learning of regular and irregular categories. In S. Thrun S. Becker and K. Obermayer, editors, *Advances in Neural Information Processing Systems 15*, pages 35–34. MIT Press, Cambridge, MA, 2003.

[4] C. Kemp, N. D. Goodman, and J. B. Tenenbaum. Learning and using relational theories. In J.C. Platt, D. Koller, Y. Singer, and S. Roweis, editors, *Advances in Neural Information Processing Systems 20*, pages 753–760. MIT Press, Cambridge, MA, 2008.

[5] N. D. Goodman, J. B. Tenenbaum, J. Feldman, and T. L. Griffiths. A rational analysis of rule-based concept learning. *Cognitive Science*, 32(1):108–154, 2008.

[6] R. N. Shepard, C. I. Hovland, and H. M. Jenkins. Learning and memorization of classifications. *Psychological Monographs*, 75(13), 1961. Whole No. 517.

[7] R. M. Nosofsky, M. Gluck, T. J. Palmeri, S. C. McKinley, and P. Glauthier. Comparing models of rule-based classification learning: A replication and extension of Shepard, Hovland, and Jenkins (1961). *Memory and Cognition*, 22:352–369, 1994.

[8] M. D. Lee and D. J. Navarro. Extending the ALCOVE model of category learning to featural stimulus domains. *Psychonomic Bulletin and Review*, 9(1):43–58, 2002.

[9] C. D. Aitkin and J. Feldman. Subjective complexity of categories defined over three-valued features. In R. Sun and N. Miyake, editors, *Proceedings of the 28th Annual Conference of the Cognitive Science Society*, pages 961–966. Psychology Press, New York, 2006.

[10] F. Mathy and J. Bradmetz. A theory of the graceful complexification of concepts and their learnability. *Current Psychology of Cognition*, 22(1):41–82, 2004.

[11] R. Vigo. A note on the complexity of Boolean concepts. *Journal of Mathematical Psychology*, 50:501–510, 2006.

[12] Y. Sakamoto and B. C. Love. Schematic influences on category learning and recognition memory. *Journal of Experimental Psychology: General*, 133(4):534–553, 2004.

[13] W. H. Crockett. Balance, agreement and positivity in the cognition of small social structures. In *Advances in Experimental Social Psychology, Vol 15*, pages 1–57. Academic Press, 1982.

[14] N. B. Cottrell. Heider's structural balance principle as a conceptual rule. *Journal of Personality and Social Psychology*, 31(4):713–720, 1975.

[15] H. B. Enderton. *A mathematical introduction to logic*. Academic Press, New York, 1972.

[16] G. Boole. *An investigation of the laws of thought on which are founded the mathematical theories of logic and probabilities*. 1854.

[17] B. C. Love and A. B. Markman. The nonindependence of stimulus properties in human category learning. *Memory and Cognition*, 31(5):790–799, 2003.

[18] L. von Ahn. Games with a purpose. *Computer*, 39(6):92–94, 2006.

[19] R. Snow, B. O'Connor, D. Jurafsky, and A. Ng. Cheap and fast–but is it good? Evaluating non-expert annotations for natural language tasks. In *Proceedings of the 2008 Conference on empirical methods in natural language processing*, pages 254–263. Association for Computational Linguistics, 2008.

